# Sub-Microwatt Analog VLSI Support Vector Machine for Pattern Classification and Sequence Estimation

**Shantanu Chakrabartty** and **Gert Cauwenberghs**
Department of Electrical and Computer Engineering
Johns Hopkins University, Baltimore, MD 21218
{*shantanu,gert*}*@jhu.edu*

## Abstract

An analog system-on-chip for kernel-based pattern classification and sequence estimation is presented. State transition probabilities conditioned on input data are generated by an integrated support vector machine. Dot product based kernels and support vector coefficients are implemented in analog programmable floating gate translinear circuits, and probabilities are propagated and normalized using sub-threshold current-mode circuits. A 14-input, 24-state, and 720-support vector forward decoding kernel machine is integrated on a 3mm×3mm chip in 0.5$\mu$m CMOS technology. Experiments with the processor trained for speaker verification and phoneme sequence estimation demonstrate real-time recognition accuracy at par with floating-point software, at sub-microwatt power.

## 1 Introduction

The key to attaining autonomy in wireless sensory systems is to embed pattern recognition intelligence directly at the sensor interface. Severe power constraints in wireless integrated systems incur design optimization across device, circuit, architecture and system levels [1]. Although system-on-chip methodologies have been primarily digital, analog integrated systems are emerging as promising alternatives with higher energy efficiency and integration density, exploiting the analog sensory interface and computational primitives inherent in device physics [2]. Analog VLSI has been chosen, for instance, to implement Viterbi [3] and HMM-based [4] sequence decoding in communications and speech processing.

Forward-Decoding Kernel Machines (FDKM) [5] provide an adaptive framework for general *maximum a posteriori* (MAP) sequence decoding, that avoid the need for backward recursion over the data in Viterbi and HMM-based sequence decoding [6]. At the core of FDKM is a support vector machine (SVM) [7] for large-margin trainable pattern classification, performing noise-robust regression of transition probabilities in forward sequence estimation. The achievable limits of FDKM power-consumption are determined by the number of support vectors (*i.e.,* regression templates), which in turn are determined by the complexity of the discrimination task and the signal-to-noise ratio of the sensor interface [8].

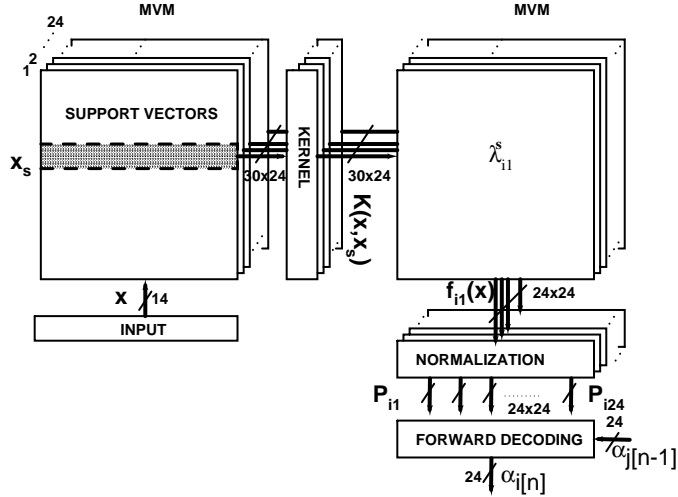

Figure 1: *FDKM system architecture.*

In this paper we describe an implementation of FDKM in silicon, for use in adaptive sequence detection and pattern recognition. The chip is fully configurable with parameters directly downloadable onto an array of floating-gate CMOS computational memory cells. By means of calibration and chip-in-loop training, the effect of mismatch and non-linearity in the analog implementation is significantly reduced.

Section 2 reviews FDKM formulation and notations. Section 3 describes the schematic details of hardware implementation of FDKM. Section 4 presents results from experiments conducted with the fabricated chip and Section 5 concludes with future directions.

## 2 FDKM Sequence Decoding

FDKM recognition and sequence decoding are formulated in the framework of MAP (maximum a posteriori) estimation, combining Markovian dynamics with kernel machines.

The MAP forward decoder receives the sequence $\overline{\mathbf{X}}[n] = \{\mathbf{x}[1], \mathbf{x}[2], \ldots, \mathbf{x}[n]\}$ and produces an estimate of conditional probability measure of state variables $q[n]$ over all classes $i \in 1, .., S$, $\alpha_i[n] = P(q[n] = i \mid \overline{\mathbf{X}}[n])$. Unlike *hidden* Markov models, the states directly encode the symbols, and the observations $\mathbf{x}$ modulate transition probabilities between states [6]. Estimates of the posterior probability $\alpha_i[n]$ are obtained from estimates of local transition probabilities using the *forward-decoding* procedure [6]

$$\alpha_i[n] = \sum_{j=1}^{S} P_{ij}[n]\, \alpha_j[n-1] \tag{1}$$

where $P_{ij}[n] = P(q[n] = i \mid q[n-1] = j, \mathbf{x}[n])$ denotes the probability of making a transition from class $j$ at time $n-1$ to class $i$ at time $n$, given the current observation vector $\mathbf{x}[n]$. Forward decoding (1) expresses first order Markovian sequential dependence of state probabilities conditioned on the data.

The transition probabilities $P_{ij}[n]$ in (1) attached to each outgoing state $j$ are obtained by normalizing the SVM regression outputs $f_{ij}(\mathbf{x})$:

$$P_{ij}[n] = [f_{ij}(\mathbf{x}[n]) - z_j[n]]_+ \tag{2}$$

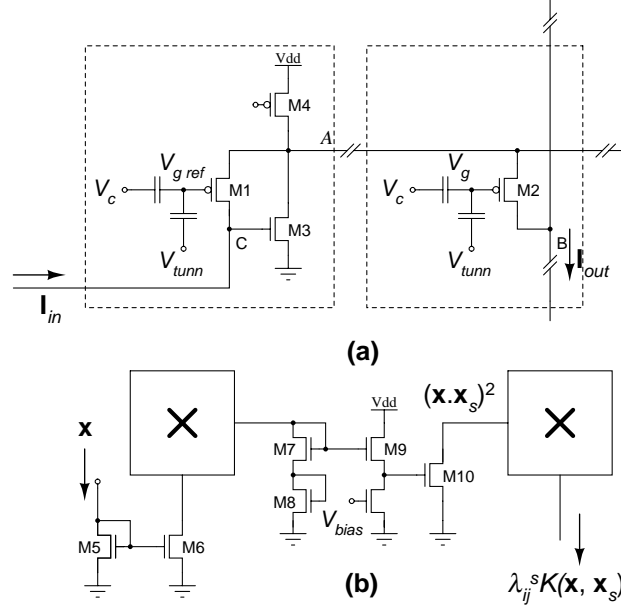

Figure 2: *Schematic of the SVM stage. (a) Multiply accumulate cell and reference cell for the MVM blocks in Figure 1. (b) Combined input, kernel and MVM modules.*

where $[.]_+ = max(.,0)$. The normalization mechanism is subtractive rather than divisive, with normalization offset factor $z_j[n]$ obtained using a reverse-waterfilling criterion with respect to a probability margin $\gamma$ [10],

$$\sum_i [f_{ij}(\mathbf{x}[n]) - z_j[n]]_+ = \gamma. \tag{3}$$

Besides improved robustness [8], the advantage of the subtractive normalization (3) is its amenability to current mode implementation as opposed to logistic normalization [11] which requires exponentiation of currents. The SVM outputs (margin variables) $f_{ij}(\mathbf{x})$ are given by:

$$f_{ij}(\mathbf{x}) = \sum_s^N \lambda_{ij}^s \, K(\mathbf{x}, \mathbf{x}_s) + b_{ij} \tag{4}$$

where $K(\cdot, \cdot)$ denotes a symmetric positive-definite kernel[1] satisfying the Mercer condition, such as a Gaussian radial basis function or a polynomial spline [7], and $\mathbf{x}_s[m], m = 1, .., N$ denote the support vectors. The parameters $\lambda_{ij}^s$ in (4) and the support vectors $\mathbf{x}_s[m]$ are determined by training on a labeled training set using a recursive FDKM procedure described in [5].

## 3  Hardware Implementation

A second order polynomial kernel $K(\mathbf{x}, \mathbf{y}) = (\mathbf{x}.\mathbf{y})^2$ was chosen for convenience of implementation. This inner-product based architecture directly maps onto an analog computational array, where storage and computation share common circuit elements. The FDKM

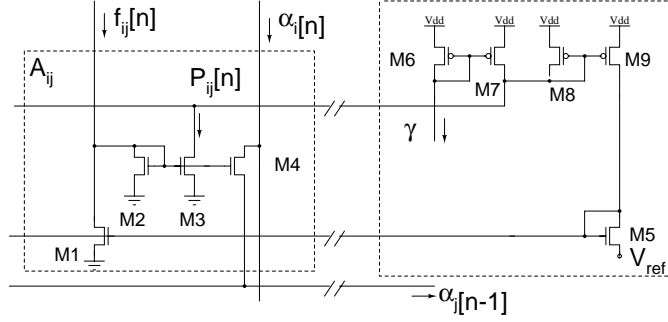

Figure 3: *Schematic of the margin propagation block.*

system architecture is shown in Figure 1. It consists of several SVM stages that generates state transition probabilities $P_{ij}[n]$ modulated by input data $\mathbf{x}[n]$, and a forward decoding block that performs *maximum a posteriori* (MAP) estimation of the state sequence $\alpha_i[n]$.

### 3.1 SVM Stage

The SVM stage implements (4) to generate unnormalized probabilities. It consists of a kernel stage computing kernels $K(\mathbf{x}_s, \mathbf{x})$ between input vector $\mathbf{x}$ and stored support vectors $\mathbf{x}_s$, and a coefficient stage linearly combining kernels using stored training parameters $\lambda_{ij}^s$. Both kernel and coefficient blocks incorporate an analog matrix-vector multiplier (MVM) with embedded storage of support vectors and coefficients. A single multiply-accumulate cell, using floating-gate CMOS non-volative analog storage, is shown in Figure 2(a). The floating gate node voltages ($V_g$) of transistors M2 are programmed using hot-electron injection and tunneling [12]. The input stage comprising transistors M1, M3 and M4 forms a key component in the design of the array and sets the voltage at node $A$ as a function of input current. By operating the array in weak-inversion, the output current through the floating gate element M2 in terms of the input stage floating gate potential $V_{gref}$ and memory element floating gate potential $V_g$ is given by

$$I_{out} = I_{in}e^{-\kappa(V_g - V_{gref})/U_T} \tag{5}$$

as a product of two pseudo-currents, leading to single quadrant multiplier. Two observations can be directly made regarding Eqn. (5):

1. The input stage eliminates the effect of the bulk on the output current, making it a function of the reference floating gate voltage which can be easily programmed for the entire row.

2. The weight is differential in the floating gate voltages $V_g - V_{gref}$, allowing to increase or decrease the weight by hot electron injection only, without the need for repeated high-voltage tunneling. For instance, the leakage current in unused rows can be reduced significantly by programming the reference gate voltage to a high value, leading to power savings.

The feedback transistor in the input stage M3 reduces the output impedance of node $A$ given by $r_o \approx g_{d1}/g_{m1}g_{m2}$. This makes the array scalable as additional memory elements can be added to the node without pulling the voltage down. An added benefit of keeping the voltage at node $A$ fixed is reduced variation in back gate parameter $\kappa$ in the floating gate elements. The current from each memory element is summed on a low impedance node established by two diode connected transistors M7-M10. This partially compensates for large Early voltage effects implicit in floating gate transistors.

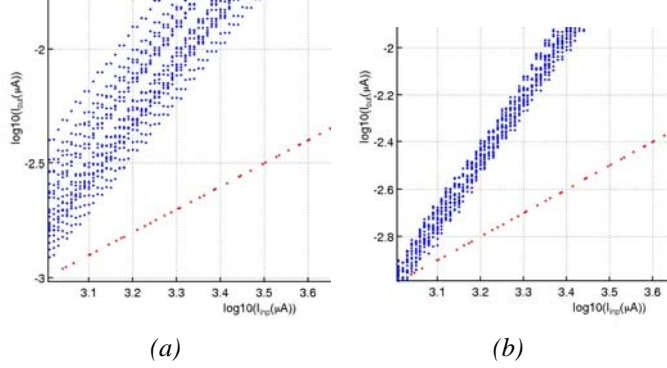

*(a)*             *(b)*

Figure 4: *Single input-output response of the SVM stage illustrating the square transfer function of the kernel block ($log(I_{out})$ vs. $log(I_{in})$) where all the MVM elements are programmed for unity gain. (a) Before calibration showing mismatch between rows. (b) After pre-distortion compensation of input and output coefficients.*

The array of elements M2 with peripheral circuits as shown in Figure 2(a) thus implement a simple single quadrant matrix-vector multiplication module. The single quadrant operation is adequate for unsigned inputs, and hence unsigned support vectors. A simple squaring circuit M7-M10 is used to implement the non-linear kernel as shown in figure 2(b). The requirement on the type of non-linearity is not stringent and can be easily incorporated into the kernel in SVM training procedure [5]. The coefficient block consists of the same matrix-vector multiplier given in figure 2(a). For the general probability model given by (2) a single quadrant multiplication is sufficient to model any distribution. This can be easily verified by observing that the distribution (2) is invariant to uniform offset in the coefficients $\lambda_{ij}^{s}$.

### 3.2 Forward Decoding Stage

The forward recursion decoding is implemented by a modified version of the sum-product probability propagation circuit in [13], performing *margin-based* probability propagation according to (1). In contrast to divisive normalization that relies on the translinear principle using sub-threshold MOS or bipolar circuits in [13], the implementation of margin-based subtractive normalization shown in figure 3 [10] is device operation independent. The circuit consists of several normalization cells $A_{ij}$ along columns computing $P_{ij} = [f_{ij} - z]_{+}$ using transistors M1-M4. Transistors M5-M9 form a feedback loop that compares and stabilizes the circuit to the normalization criterion (3). The currents through transistors M4 are auto-normalized to the previous state value $\alpha_j[n-1]$ to produce a new estimate of $\alpha_i[n1]$ based on recursion (1). The delay in equation (1) is implemented using a log-domain filter and a fixed normalization current ensures that all output currents be properly scaled to stabilize the continuous-time feedback loop.

## 4 Experimental Results

A 14-input, 24-state, and 24×30-support vector FDKM was integrated on a 3mm×3mm FDKM chip, fabricated in a $0.5\mu$m CMOS process, and fully tested. Figure 5(c) shows the micrograph of the fabricated chip. Labeled training data pertaining to a certain task were used to train an SVM, and the training coefficients thus obtained were programmed onto the chip.

Table 1: FDKM Chip Summary

| Technology | Value |
| --- | --- |
| Area | 3mm×3mm |
| Technology | $0.5\mu$ CMOS |
| Supply Voltage | 4 V |
| **System Parameters** | |
| Floating Cell Count | 28814 |
| Number of Support Vectors | 720 |
| Input Dimension | 14 |
| Number of States | 24 |
| Power Consumption | 80nW - 840nW |
| Energy Efficiency | 1.6pJ/MAC |

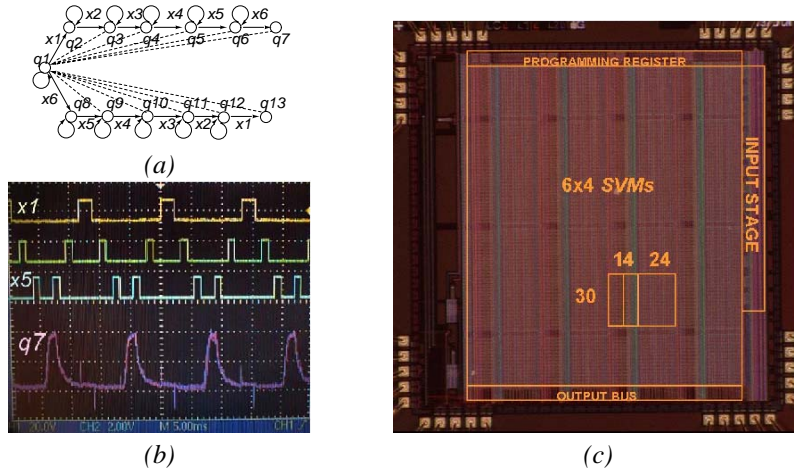

Figure 5: *(a) Transition-based sequence detection in a 13-state Markov model. (b) Experimental recording of $\alpha_7 = P(q_7)$, detecting one of two recurring sequences in inputs $x_1 \rightarrow x_6$ ($x_1$, $x_3$ and $x_5$ shown). (c) Micrograph of the FDKM chip*

Programming of the trained coefficients was performed by programming respective cells M2 along with the corresponding input stage M1, so as to establish the desired ratio of currents. The values were established by continuing hot electron injection until the desired current was attained. During hot electron injection, the control gate $V_c$ was adjusted to set the injection current to a constant level for stable injection. All cells in the kernel and coefficient modules of the SVM stage are random accessible for read, write and calibrate operations. The calibration procedure compensates for mismatch between different input/output paths by adapting the floating gate elements in the MVM cells. This is illustrated in Figure 4 where the measured square kernel transfer function is shown before and after calibration.

The chip is fully reconfigurable and can perform different recognition tasks by programming different training parameters, as demonstrated through three examples below. Depending on the number of active support vectors and the absolute level of currents (in relation to decoding bandwidth), power dissipation is in the lower nanowatt to microwatt range.

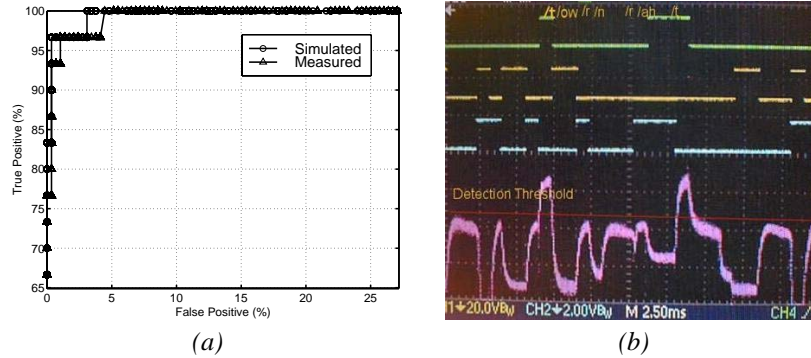

*(a)*                                        *(b)*

Figure 6: *(a) Measured and simulated ROC curve for the speaker verification experiment. (b) Experimental phoneme recognition by FDKM chip. The state probability shown is for consonant $/t/$ in words "torn," "rat," and "error." Two peaks are located as expected from the input sequence, shown on top.*

For the first set of experiments, parameters corresponding to a simple Markov chain shown in figure 5(a) were programmed onto the chip to differentiate between two given sequences of input features: one a sweep of active input components in rising order ($x_1$ through $x_6$), and the other in descending order ($x_6$ through $x_1$). The output of state $q_7$ in the Markov chain is shown in figure 5(b). It can be clearly observed that state $q_7$ "fires" only when a rising sequence of pulse trains arrives. The FDKM chip thereby demonstrates probability propagation similar to that in the architecture of [4]. The main difference is that the present architecture can be configured for detecting other, more complex sequences through programming and training.

For the second set of experiments the FDKM chip was programmed to perform speaker verification using speech data from YOHO corpus. For training we chose 480 utterances corresponding to 10 separate speakers (101-110). For each of these utterances 12 mel-cepstra coefficients were computed for every 25ms frames. These coefficients were clustered using k-means clustering to obtain 50 clusters per speaker which were then used for training the SVM. For testing 480 utterances for those speakers were chosen, and confidence scores returned by the SVMs were integrated over all frames of an utterance to obtain a final decision. Verification results obtained from the chip demonstrate 97% true acceptance at 1% false positive rate, identical to the performance obtained through floating point software simulations as shown by the receiver operating characteristic shown in figure 6(a). The total power consumption for this task is only 840nW, demonstrating its suitability for autonomous sensor applications.

A third set of experiment aimed at detecting phone utterances in human speech. Mel-cepstra coefficients of six phone utterances ($/t/,/n/,/r/,/ow/,/ah/,/eh/$) selected from the TIMIT corpus were transformed using singular value decomposition and thresholding. Even though the recognition was demonstrated for the reduced set of features, the chip operates internally with analog inputs. Figure 6(b) illustrates correct detection of phonemes as identified by the presence of phone $/t/$ at the expected time instances in the input sequence.

## 5 Discussion and Conclusion

We designed an FDKM based sequence recognition system on silicon and demonstrated its performance on simple but general tasks. The chip is fully reconfigurable and different sequence recognition engines can be programmed using parameters obtained through

SVM training. FDKM decoding is performed in real-time and is ideally suited for sequence recognition and verification problems involving speech features. All analog processing in the chip is performed by transistors operating in weak-inversion resulting in power dissipation in the nanowatt to microwatt range. Non-volatile storage of training parameters further reduces standby power dissipation.

We also note that while low power dissipation is a virtue in many applications, increased power can be traded for increased bandwidth. For instance, the presented circuits could be adapted using heterojunction bipolar junction transistors in a SiGe process for ultra-high speed MAP decoding applications in digital communication, using essentially the same FDKM architecture as presented here.

**Acknowledgement:** This work is supported by a grant from The Catalyst Foundation (*http://www.catalyst-foundation.org*), NSF IIS-0209289, ONR/DARPA N00014-00-C-0315, and ONR N00014-99-1-0612. The chip was fabricated through the MOSIS service.

## Footnotes

[1]$K(\mathbf{x}, \mathbf{y}) = \Phi(\mathbf{x}).\Phi(\mathbf{y})$. The map $\Phi(\cdot)$ need not be computed explicitly, as it only appears in inner-product form.

# References

[1] Wang, A. and Chandrakasan, A.P, "Energy-Efficient DSPs for Wireless Sensor Networks," IEEE Signal Proc. Mag., vol. 19 (4), pp. 68-78, July 2002.

[2] Vittoz, E.A., "Low-Power Design: Ways to Approach the Limits," *Dig. 41st IEEE Int. Solid-State Circuits Conf. (ISSCC)*, San Francisco CA, 1994.

[3] Shakiba, M.S, Johns, D.A, and Martin, K.W, "BiCMOS Circuits for Analog Viterbi Decoders," *IEEE Trans. Circuits and Systems II*, vol. 45 (12), Dec. 1998.

[4] Lazzaro, J, Wawrzynek, J, and Lippmann, R.P, "A Micropower Analog Circuit Implementation of Hidden Markov Model State Decoding," *IEEE J. Solid-State Circuits,* vol. 32 (8), Aug. 1997.

[5] Chakrabartty, S. and Cauwenberghs, G. "Forward Decoding Kernel Machines: A hybrid HMM/SVM Approach to Sequence Recognition," *IEEE Int. Conf. of Pattern Recognition: SVM workshop. (ICPR'2002)*, Niagara Falls, 2002.

[6] Bourlard, H. and Morgan, N., *Connectionist Speech Recognition: A Hybrid Approach,* Kluwer Academic, 1994.

[7] Vapnik, V. *The Nature of Statistical Learning Theory,* New York: Springer-Verlag, 1995.

[8] Chakrabartty, S., and Cauwenberghs, G. "Power Dissipation Limits and Large Margin in Wireless Sensors," *Proc. IEEE Int. Symp. Circuits and Systems(ISCAS2003)*, vol. 4, 25-28, May 2003.

[9] Bahl, L.R., Cocke J., Jelinek F. and Raviv J. "Optimal Decoding of Linear Codes for Minimizing Symbol Error Rate," *IEEE Transactions on Inform. Theory*, vol. **IT-20**, pp. 284-287, 1974.

[10] Chakrabartty, S., and Cauwenberghs, G. "Margin Propagation and Forward Decoding in Analog VLSI," *Proc. IEEE Int. Symp. Circuits and Systems(ISCAS2004)*, Vancouver Canada, May 23-26, 2004.

[11] Jaakkola, T. and Haussler, D. "Probabilistic kernel regression models," *Proc. Seventh Int. Workshop Artificial Intelligence and Statistics* , 1999.

[12] C. Dorio,P. Hasler,B. Minch and C.A. Mead, "A Single-Transistor Silicon Synapse," *IEEE Trans. Electron Devices,* vol. 43 (11), Nov. 1996.

[13] H. Loeliger, F. Lustenberger, M. Helfenstein and F. Tarkoy, "Probability Propagation and Decoding in Analog VLSI," *IEEE Proc. ISIT*, 1998.